# A Silicon Model of Amplitude Modulation Detection in the Auditory Brainstem

**André van Schaik, Eric Fragnière, Eric Vittoz**
MANTRA Center for Neuromimetic Systems
Swiss Federal Institute of Technology
CH-1015 Lausanne
email: Andre.van_Schaik@di.epfl.ch

## Abstract

Detection of the periodicity of amplitude modulation is a major step in the determination of the pitch of a sound. In this article we will present a silicon model that uses synchronicity of spiking neurons to extract the fundamental frequency of a sound. It is based on the observation that the so called 'Choppers' in the mammalian Cochlear Nucleus synchronize well for certain rates of amplitude modulation, depending on the cell's intrinsic chopping frequency. Our silicon model uses three different circuits, i.e., an artificial cochlea, an Inner Hair Cell circuit, and a spiking neuron circuit.

## 1. INTRODUCTION

Over the last few years, we have developed and implemented several analog VLSI building blocks that allow us to model parts of the auditory pathway [1], [2], [3]. This paper presents one experiment using these building blocks to create a model for the detection of the fundamental frequency of a harmonic complex. The estimation of this fundamental frequency by the model shows some important similarities with psychoacoustic experiments in pitch estimation in humans [4]. A good model of pitch estimation will give us valuable insights in the way the brain processes sounds. Furthermore, a practical application to speech recognition can be expected, either by using the pitch estimate as an element in the acoustic vector fed to the recognizer, or by normalizing the acoustic vector to the pitch.

Although the model doesn't yield a complete model of pitch estimation, and explains probably only one of a few different mechanisms the brain uses for pitch estimation, it can give us a better understanding of the physiological background of psycho-acoustic results. An electronic model can be especially helpful, when the parameters of the model can be easily controlled, and when the model will operate in real time.

## 2. THE MODEL

The model was originally developed by Hewitt and Meddis [4], and was based on the observation that Chopper cells in the Cochlear Nucleus synchronize when the stimulus is modulated in amplitude within a particular modulation frequency range [5].

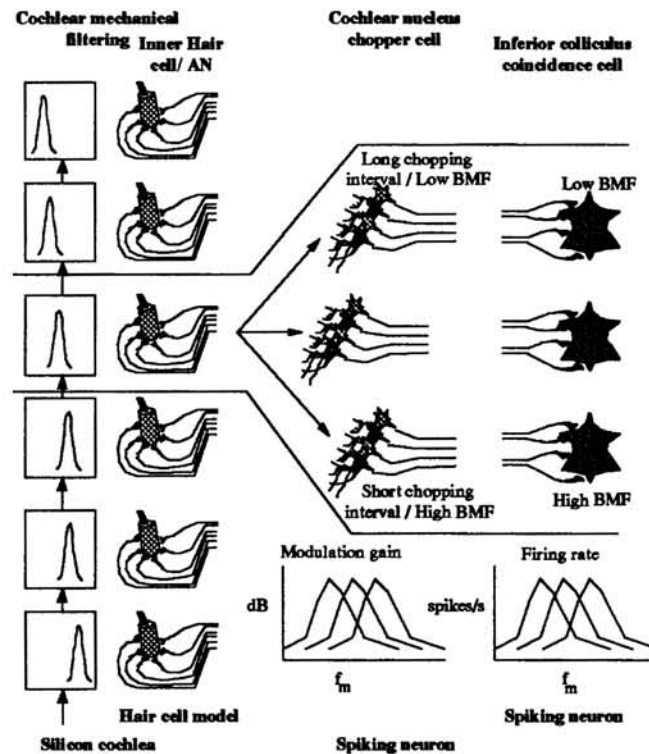

Fig. 1. Diagram of the AM detection model. BMF=Best Modulation Frequency.

The diagram shown in figure 1 shows the elements of the model. The cochlea filters the incoming sound signal. Since the width of the pass-band of a cochlear band-pass filter is proportional to its cut-off frequency, the filters will not be able to resolve the individual harmonics of a high frequency carrier (>3kHz) amplitude modulated at a low rate (<500Hz). The outputs of the cochlear filters that have their cut-off frequency slightly above the carrier frequency of the signal will therefore still be modulated in amplitude at the original modulation frequency. This modulation component will therefore synchronize a certain group of Chopper cells. The synchronization of this group of Chopper cells can be detected using a coincidence detecting neuron, and signals the presence of a particular amplitude modulation frequency. This model is biologically plausible, because it is known that the choppers synchronize to a particular amplitude modulation frequency and that they project their output towards the Inferior Colliculus (amongst others). Furthermore, neurons that can function as coincidence detectors are shown to be present in the Inferior Colliculus and the rate of firing of these neurons is a

band-pass function of the amplitude modulation rate. It is not known to date however if the choppers actually project to these coincidence detector neurons.

The actual mechanism that synchronizes the chopper cells will be discussed with the measurements in section 4. In the next section, we will first present the circuits that allowed us to build the VLSI implementation of this model.

## 3. THE CIRCUITS

All of the circuits used in our model have already been presented in more detail elsewhere, but we will present them briefly for completeness. Our silicon cochlea has been presented in detail at NIPS'95 [1], and more details about the Inner Hair Cell circuit and the spiking neuron circuit can be found in [2].

### 3.1 THE SILICON COCHLEA

The silicon cochlea consists of a cascade of second order low-pass filters. Each filter section is biased using Compatible Lateral Bipolar Transistors (CLBTs) which control the cut-off frequency and the quality factor of each section. A single resistive line is used to bias all CLBTs. Because of the exponential relation between the Base-Emitter Voltage and the Collector current of the CLBTs, the linear voltage gradient introduced by the resistive line will yield a filter cascade with an exponentially decreasing cut-off frequency of the filters. The output voltage of each filter $V_{out}$ then represents the displacement of a basilar membrane section. In order to obtain a representation of the basilar membrane velocity, we take the difference between $V_{out}$ and the voltage on the internal node of the second order filter.

We have integrated this silicon cochlea using 104 filter stages, and the output of every second stage is connected to an output pin.

### 3.2 THE INNER HAIR CELL MODEL

The inner hair cell circuit is used to half-wave rectify the basilar membrane velocity signal and to perform some form of temporal adaptation, as can be seen in figure 2b. The differential pair at the input is used to convert the input voltage into a current with a compressive relation between input amplitude and the actual amplitude of the current.

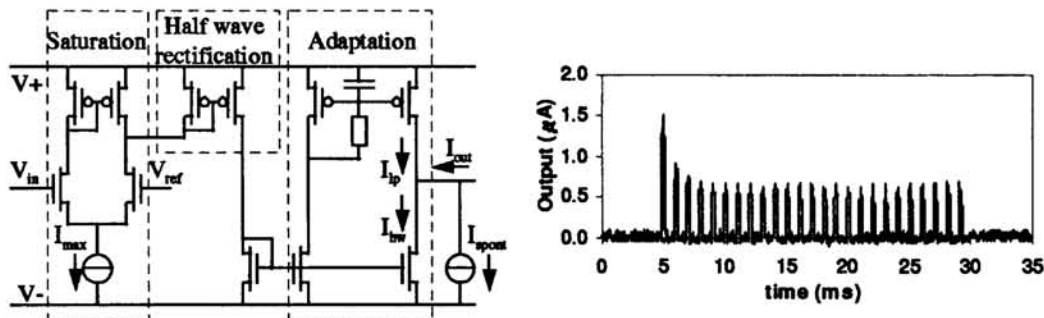

Fig. 2. a) The Inner Hair Cell circuit, b) measured output current.

We have integrated a small chip containing 4 independent inner hair cells.

### 3.3 THE SPIKING NEURON MODEL

The spiking neuron circuit is given in figure 3. The membrane of a biological neuron is modeled by a capacitance, $C_{mem}$, and the membrane leakage current is controlled by the gate voltage, $V_{leak}$, of an NMOS transistor. In the absence of any input ($I_{ex}=0$), the membrane voltage will be drawn to its resting potential (controlled by $V_{rest}$), by this leakage current. Excitatory inputs simply add charge to the membrane capacitance, whereas inhibitory inputs are simply modeled by a negative $I_{ex}$. If an excitatory current larger than the leakage current of the membrane is injected, the membrane potential will increase from its resting potential. This membrane potential, $V_{mem}$, is compared with a controllable threshold voltage $V_{thres}$, using a basic transconductance amplifier driving a high impedance load. If $V_{mem}$ exceeds $V_{thres}$, an action potential will be generated.

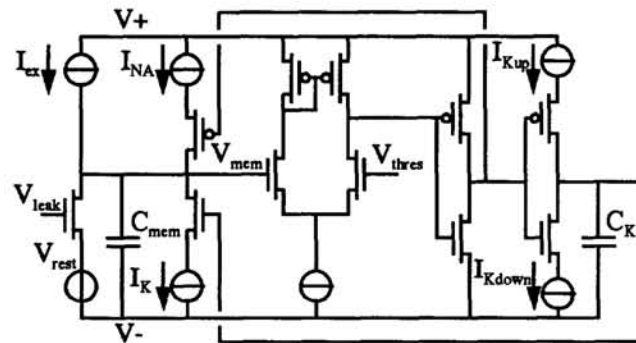

Fig. 3. The Spiking Neuron circuit

The generation of the action potential happens in a similar way as in the biological neuron, where an increased sodium conductance creates the upswing of the spike, and a delayed increase of the potassium conductance creates the downswing. In the circuit this is modeled as follows. If $V_{mem}$ rises above $V_{thres}$, the output voltage of the comparator will rise to the positive power supply. The output of the following inverter will thus go low, thereby allowing the "sodium current" $I_{Na}$ to pull up the membrane potential. At the same time however, a second inverter will allow the capacitance $C_K$ to be charged at a speed which can be controlled by the current $I_{Kup}$. As soon as the voltage on $C_K$ is high enough to allow conduction of the NMOS to which it is connected, the "potassium current" $I_K$ will be able to discharge the membrane capacitance.

If $V_{mem}$ now drops below $V_{thres}$, the output of the first inverter will become high, cutting off the current $I_{Na}$. Furthermore, the second inverter will then allow $C_K$ to be discharged by the current $I_{Kdown}$. If $I_{Kdown}$ is small, the voltage on $C_K$ will decrease only slowly, and, as long as this voltage stays high enough to allow $I_K$ to discharge the membrane, it will be impossible to stimulate the neuron if $I_{ex}$ is smaller than $I_K$. Therefore $I_{Kdown}$ can be said to control the 'refractory period' of the neuron.

We have integrated a chip, containing a group of 32 neurons, each having the same bias voltages and currents. The component mismatch and the noise ensure that we actually have 32 similar, but not completely equal neurons.

## 4. TEST RESULTS

Most neuro-physiological data concerning low frequency amplitude modulation of high frequency carriers exists for carriers at about 5kHz and a modulation depth of about 50%. We therefore used a 5 kHz sinusoid in our tests and a 50% modulation depth at frequencies below 550Hz.

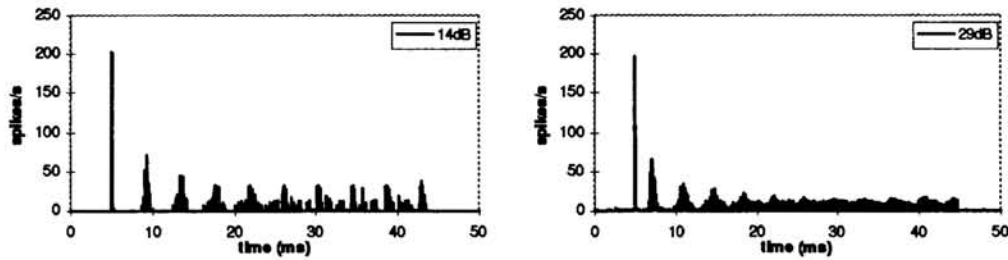

Fig. 4. PSTH of the chopper chip for 2 different sound intensities

First step in the elaboration of the model is to test if the group of spiking neurons on a single chip is capable of performing like a group of similar Choppers. Neurons in the auditory brainstem are often characterized with a Post Stimulus Time Histogram (PSTH), which is a histogram of spikes in response to repeated stimulation with a pure tone of short duration. If the choppers on the chip are really similar, the PSTH of this group of choppers will be very similar to the PSTH of a single chopper. In figure 4 the PSTH of the circuit is shown. It is the result of the summed response of the 32 neurons on chip to 20 repeated stimulations with a 5kHz tone burst. This figure shows that the response of the Choppers yields a PSTH typical of chopping neurons, and that the chopping frequency, keeping all other parameters constant, increases with increasing sound intensity. The chopping rate for an input signal of given intensity can be controlled by setting the refractory period of the spiking neurons, and can thus be used to create the different groups of choppers shown in figure 1. The chopping rate of the choppers in figure 4 is about 300Hz for a 29dB input signal.

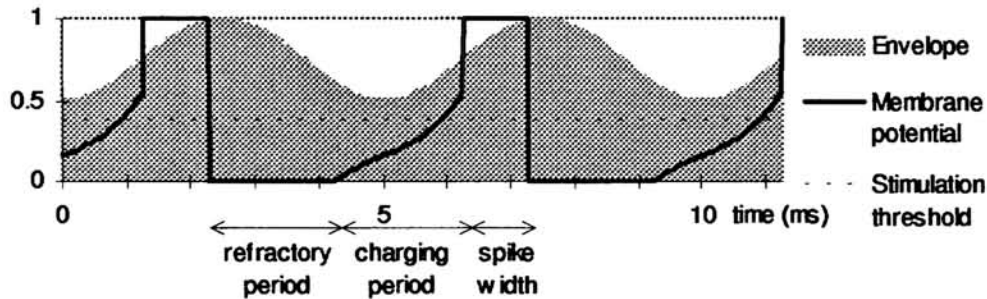

Fig. 5. Spike generation for a Chopper cell.

To understand why the Choppers will synchronize for a certain amplitude modulation frequency, one has to look at the signal envelope, which contains temporal information on a time scale that can influence the spiking neurons. The 5kHz carrier itself will not contain any temporal information that influences the spiking neuron in an important way. Consider the case when the modulation frequency is similar to the chopping frequency (figure 5). If a Chopper then spikes during the rising flank of the envelope, it will come out of its refractory period just before the next rising flank of the envelope. If the driven chopping frequency is a bit too low, the Chopper will come out of its refractory period a bit later, therefore it receives a higher average stimulation and it spikes a little higher on the rising flank of the envelope. This in turn increases the chopping frequency and thus provides a form of negative feedback on the chopping frequency. This therefore makes spiking on a certain point on the rising flank of the envelope a stable situation. With the same reasoning one can show that spiking on the falling flank is therefore an unstable situation. Furthermore, it is not possible to stabilize a cell driven above its maximum chopping rate, nor is it possible to stabilize a cell that fires more than once per modulation period. Since a group of similar choppers will

stabilize at about the same point on the rising flank, their spikes will thus coincide when the modulation frequency allows them to.

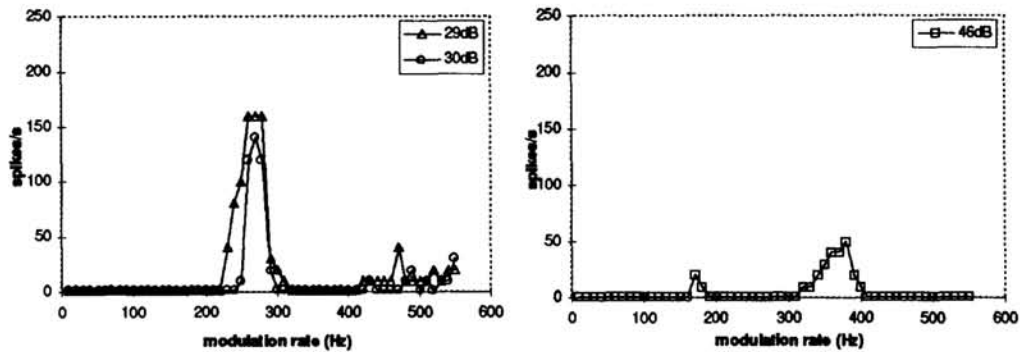

Fig. 6. AM sensitivity of the coincidence detecting neuron.

Another free parameter of the model is the threshold of the coincidence detecting neuron. If this parameter is set so that at least 60% of the choppers must spike within 1ms to be considered a coincidence, we obtain the output of figure 6. We can see that this yields the expected band-pass Modulation Transfer Function (MTF), and that the best modulation frequency for the 29dB input signal corresponds to the intrinsic chopping rate of the group of neurons. Figure 6 also shows that the best modulation frequency (BMF), just as the chopping rate, increases with increasing sound intensity, but that the maximum number of spikes per second actually decreases. This second effect is caused by the fact that the stabilizing effect of the positive flank of the signal envelope only influences the time during which the neuron is being charged, which becomes a smaller part of the total spiking period at higher intensities. The negative feedback thus has less influence on the total chopping period and therefore synchronizes the choppers less.

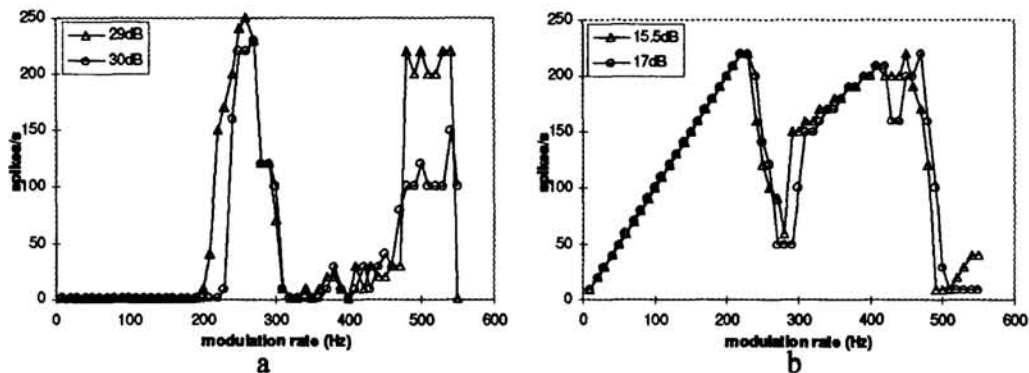

a                                         b

Fig. 7. AM sensitivity of the coincidence detecting neuron.

When the coincidence threshold is lowered to 50%, we can see in figure 7a that the maximum number of spikes goes up, because this threshold is more easily reached. Furthermore, a second pass-band shows up at double the BMF. This is because the choppers fire only every second amplitude modulation period, and part of the group of choppers will synchronize during the odd periods, whereas others during the even periods. The division of the group of choppers will typically be close to, but hardly ever exactly 50-50, so that either during the odd or during the even modulation period the 50% coincidence threshold is exceeded. The 60% threshold of figure 6 will only rarely be exceeded, explaining the weak second peak seen around 500Hz in this figure.

Figure 7b. shows the MTF for low intensity signals with a 50% coincidence threshold. At low intensities the effect of an additional non-linearity, the stimulation threshold, shows up. Whenever the instantaneous value of the envelope is lower than the stimulation threshold, the spiking neuron will not be stimulated because its input current will be lower than the cell's leakage current. At these low intensities the activity during the valleys of the modulation envelope will thus not be enough to stimulate the Choppers (see figure 5). For stimuli with a lower modulation frequency than the group's chopping frequency, the Choppers will come out of their refractory period in such a valley. These choppers therefore will have to wait for the envelope amplitude to increase above a certain value, before they receive anew a stimulation. This waiting period nullifies the effect of the variation of the refractory period of the Choppers, and thus synchronizes the Choppers for low modulation frequencies. A second effect of this waiting period is that in this case the firing rate of the Choppers matches the modulation frequency. When the modulation frequency becomes higher than the maximum chopping frequency, the Choppers will fire only every second period, but will still be synchronized, as can be seen between 300Hz and 500Hz in figure 7b.

## 5. CONCLUSIONS

In this article we have shown that it is possible to use our building blocks to build a multi-chip system that models part of the auditory pathway. Furthermore, the fact that the spiking neuron chip can be easily biased to function as a group of similar Choppers, combined with the relative simplicity of the spike generation mechanism of a single neuron on chip, allowed us to gain insight in the process by which chopping neurons in the mammalian Cochlear Nucleus synchronize to a particular range of amplitude modulation frequencies.

### References

[1] A. van Schaik, E. Fragnière, & E. Vittoz, "Improved silicon cochlea using compatible lateral bipolar transistors," *Advances in Neural Information Processing Systems 8*, MIT Press, Cambridge, 1996.

[2] A. van Schaik, E. Fragnière, & E. Vittoz, "An analoge electronic model of ventral cochlear nucleus neurons," *Proc. Fifth Int. Conf. on Microelectronics for Neural Networks and Fuzzy Systems*, IEEE Computer Society Press, Los Alamitos, 1996, pp. 52-59.

[3] A. van Schaik and R. Meddis, "The electronic ear; towards a blueprint," *Neurobiology*, NATO ASI series, Plenum Press, New York, 1996.

[4] M.J. Hewitt and R. Meddis, "A computer model of amplitude-modulation sensitivity of single units in the inferior colliculus," *J. Acoust. Soc. Am.*, 95, 1994, pp. 1-15.

[5] M.J. Hewitt, R. Meddis, & T.M. Shackleton, "A computer model of a cochlear-nucleus stellate cell: responses to amplitude-modulated and pure tone stimuli," *J. Acoust. Soc. Am.*, 91, 1992, pp. 2096-2109.
